# A Comparative Study of the Practical Characteristics of Neural Network and Conventional Pattern Classifiers

Kenney Ng
BBN Systems and Technologies
Cambridge, MA 02138

Richard P. Lippmann
Lincoln Laboratory, MIT
Lexington, MA 02173-9108

## Abstract

Seven different pattern classifiers were implemented on a serial computer and compared using artificial and speech recognition tasks. Two neural network (radial basis function and high order polynomial GMDH network) and five conventional classifiers (Gaussian mixture, linear tree, $K$ nearest neighbor, KD-tree, and condensed $K$ nearest neighbor) were evaluated. Classifiers were chosen to be representative of different approaches to pattern classification and to complement and extend those evaluated in a previous study (Lee and Lippmann, 1989). This and the previous study both demonstrate that classification error rates can be equivalent across different classifiers when they are powerful enough to form minimum error decision regions, when they are properly tuned, and when sufficient training data is available. Practical characteristics such as training time, classification time, and memory requirements, however, can differ by orders of magnitude. These results suggest that the selection of a classifier for a particular task should be guided not so much by small differences in error rate, but by practical considerations concerning memory usage, computational resources, ease of implementation, and restrictions on training and classification times.

## 1  INTRODUCTION

Few studies have compared practical characteristics of adaptive pattern classifiers using real data. There has frequently been an over-emphasis on back-propagation classifiers and artificial problems and a focus on classification error rate as the main performance measure. No study has compared the practical trade-offs in training time, classification time, memory requirements, and complexity provided by the

many alternative classifiers that have been developed (e.g. see Lippmann 1989).

The purpose of this study was to better understand and explore practical character-istics of classifiers not included in a previous study (Lee and Lippmann, 1989; Lee 1989). Seven different neural network and conventional pattern classifiers were eval-uated. These included radial basis function (RBF), high order polynomial GMDH network, Gaussian mixture, linear decision tree, $K$ nearest neighbor (KNN), KD tree, and condensed $K$ nearest neighbor (CKNN) classifiers. All classifiers were implemented on a serial computer (Sun 3-110 Workstation with FPA) and tested using a digit recognition task (7 digits, 22 cepstral inputs, 16 talkers, 70 training and 112 testing patterns per talker), a vowel recognition task (10 vowels, 2 formant frequency inputs, 67 talkers, 338 training and 333 testing patterns), and two ar-tificial tasks with two input dimensions that require either a single convex or two disjoint decision regions. Tasks are as in (Lee and Lippmann, 1989) and details of experiments are described in (Ng, 1990).

## 2    TUNING EXPERIMENTS

Internal parameters or weights of classifiers were determined using training data. Global free parameters that provided low error rates were found experimentally using cross-validation and the training data or by using test data. Global parameters included an overall basis function width scale factor for the RBF classifier, order of nodal polynomials for the GMDH network, and number of nearest neighbors for the KNN, KD tree, and CKNN classifiers.

Experiments were also performed to match the complexity of each classifier to that of the training data. Many classifiers exhibit a characteristic divergence between training and testing error rates as a function of their complexity. Poor performance results when a classifier is too simple to model the complexity of training data and also when it is too complex and "over-fits" the training data. Cross-validation and statistical techniques were used to determine the correct size of the linear tree and GMDH classifiers where training and test set error rates diverged substantially. An information theoretic measure (Predicted Square Error) was used to limit the complexity of the GMDH classifier. This classifier was allowed to grow by adding layers and widening layers to find the number of layers and the layer width which minimized predicted square error. Nodes in the linear tree were pruned using 10-fold cross-validation and a simple statistical test to determine the minimum size tree that provides good performance. Training and test set error rates did not diverge for the RBF and Gaussian mixture classifiers. Test set performance was thus used to determine the number of Gaussian centers for these classifiers.

A new multi-scale radial basis function classifier was developed. It has multiple radial basis functions centered on each basis function center with widths that vary over 1 1/2 orders of magnitude. Multi-scale RBF classifiers provided error rates that were similar to those of more conventional RBF classifiers but eliminated the need to search for a good value of the global basis function width scale factor.

The CKNN classifier used in this study was also new. It was developed to reduce memory requirements and dependency on training data order. In the more conven-tional CKNN classifier, training patterns are presented sequentially and classified using a KNN rule. Patterns are stored as exemplars only if they are classified in-

correctly. In the new CKNN classifier, this conventional CKNN training procedure is repeated N times with different orderings of the training patterns. All exemplar patterns stored using any ordering are combined into a new reduced set of training patterns which is further pruned by using it as training data for a final pass of conventional CKNN training. This approach typically required less memory than a KNN or a conventional CKNN classifier. Other experiments described in (Chang and Lippmann, 1990) demonstrate how genetic search algorithms can further reduce KNN classifier memory requirements.

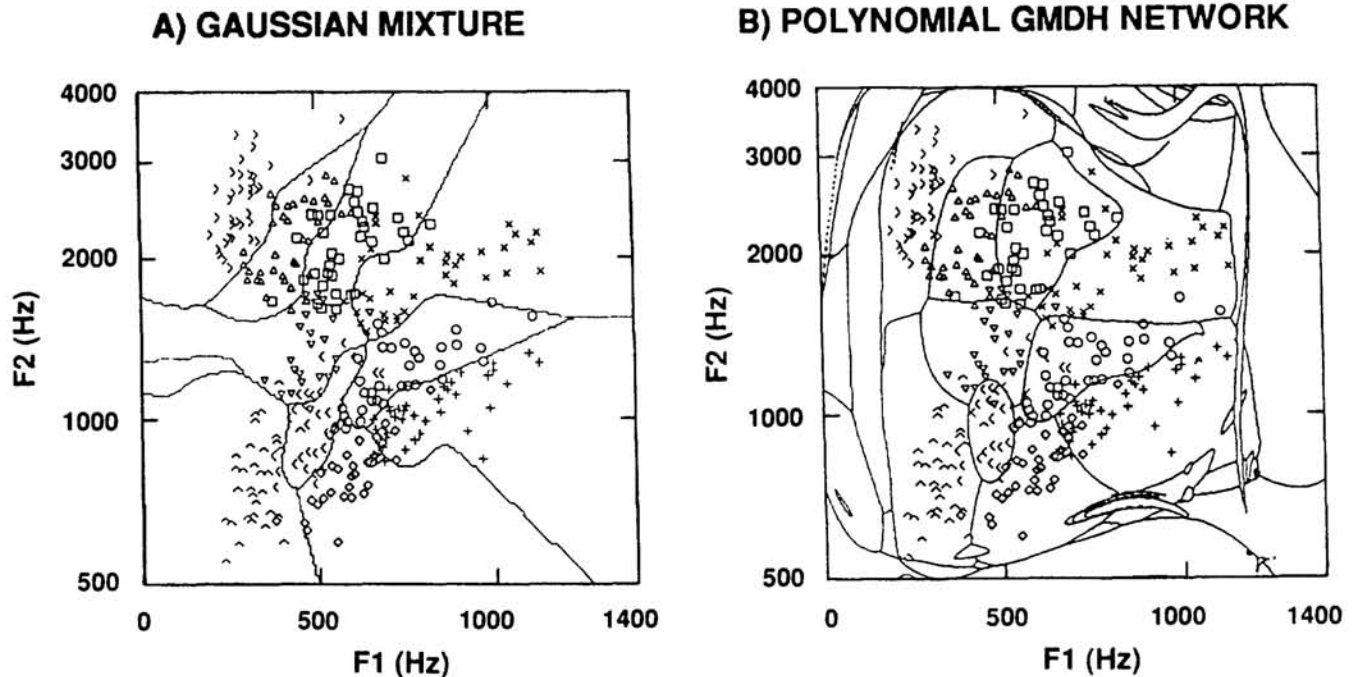

Figure 1: Decision Regions Created by (A) RBF and (B) GMDH Classifiers for the Vowel Problem.

## 3   DECISION REGIONS

Classifiers differ not only in their structure and training but also in how decision regions are formed. Decision regions formed by the RBF classifier for the vowel problem are shown in Figure 1A. Boundaries are smooth spline-like curves that can form arbitrarily complex regions. This improves generalization for many real problems where data for different classes form one or more roughly ellipsoidal clusters. Decision regions for the high-order polynomial (GMDH) network classifier are shown in Figure 1B. Decision region boundaries are smooth and well behaved only in regions of the input space that are densely sampled by the training data. Decision boundaries are erratic in regions where there is little training data due to the high polynomial order of the discriminant functions formed by the GMDH classifier. As a result, the GMDH classifier generalizes poorly in regions with little training data. Decision boundaries for the linear tree classifier are hyperplanes. This classifier may also generalize poorly if data is in ellipsoidal clusters.

## 4   ERROR RATES

Figure 2 shows the classification (test set) error rates for all classifiers on the bulls-eye, disjoint, vowel, and digit problems. The solid line in each plot represents the

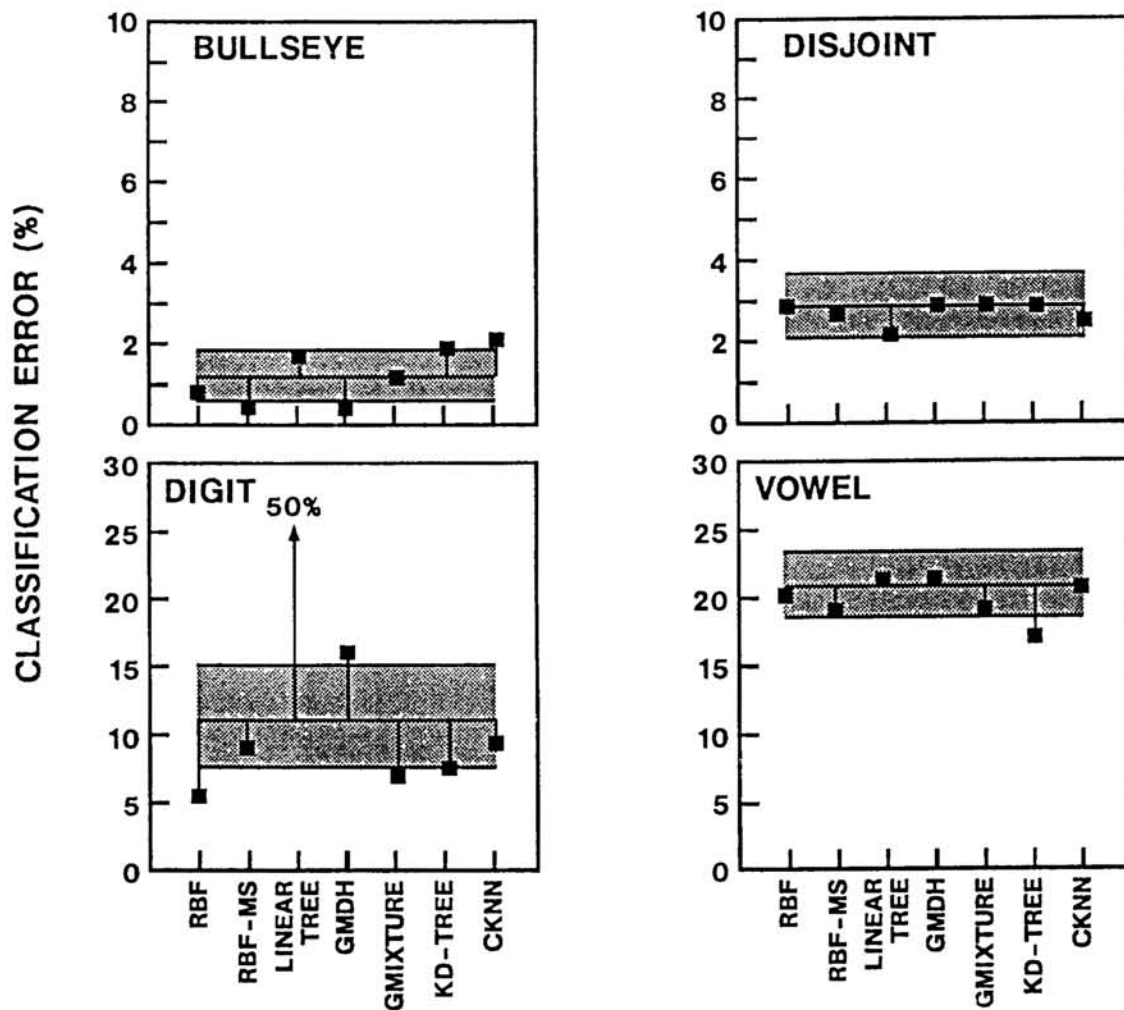

Figure 2: Test Data Error Rates for All Classifiers and All Problems.

mean test set error rate across all the classifiers for that problem. The shaded regions represent one binomial standard deviation, $\sigma$, above and below. The binomial standard deviation was calculated as $\sigma = \sqrt{\mathcal{E}(1 - \mathcal{E})/N}$, where $\mathcal{E}$ is the estimated average problem test set error rate and $N$ is the number of test patterns for each problem. The shaded region gives a rough measure of the range of expected statistical fluctuation if the error rates for different classifiers are identical. A more detailed statistical analysis of the test set error rates for classifiers was performed using McNemar's significance test. At a significance level of $\alpha = 0.01$, the error rates of the different classifiers on the bullseye, disjoint, and vowel problems do *not* differ significantly from each other.

Performance on the more difficult digit problem, however, did differ significantly across classifiers. This problem has very little training data (10 training patterns per class) and high dimensional inputs (an input dimension of 22). Some classifiers, including the RBF and Gaussian mixture classifiers, were able to achieve very low error rates on this problem and generalize well even in this high dimensional space with little training data. Other classifiers, including the multi-scale RBF, KD-tree, and CKNN classifiers, provided intermediate error rates. The GMDH network classifier and the linear tree classifier provided high error rates.

The linear tree classifier performed poorly on the digit problem because there is

not enough training data to sample the input space densely enough for the training algorithm to form decision boundaries that can generalize well. The poor performance of the GMDH network classifier is due, in part, to the inability of the GMDH network classifier to extrapolate well to regions with little training data.

## 5    PERFORMANCE TRADE-OFFS

Although differences in the error rates of most classifiers are small, differences in practical performance characteristics are often large. For example, on the vowel problem, although both the Gaussian mixture and KD tree classifiers perform well, the Gaussian mixture classifier requires 20 times *less* classification memory than the KD tree classifier, but takes 10 times *longer* to train.

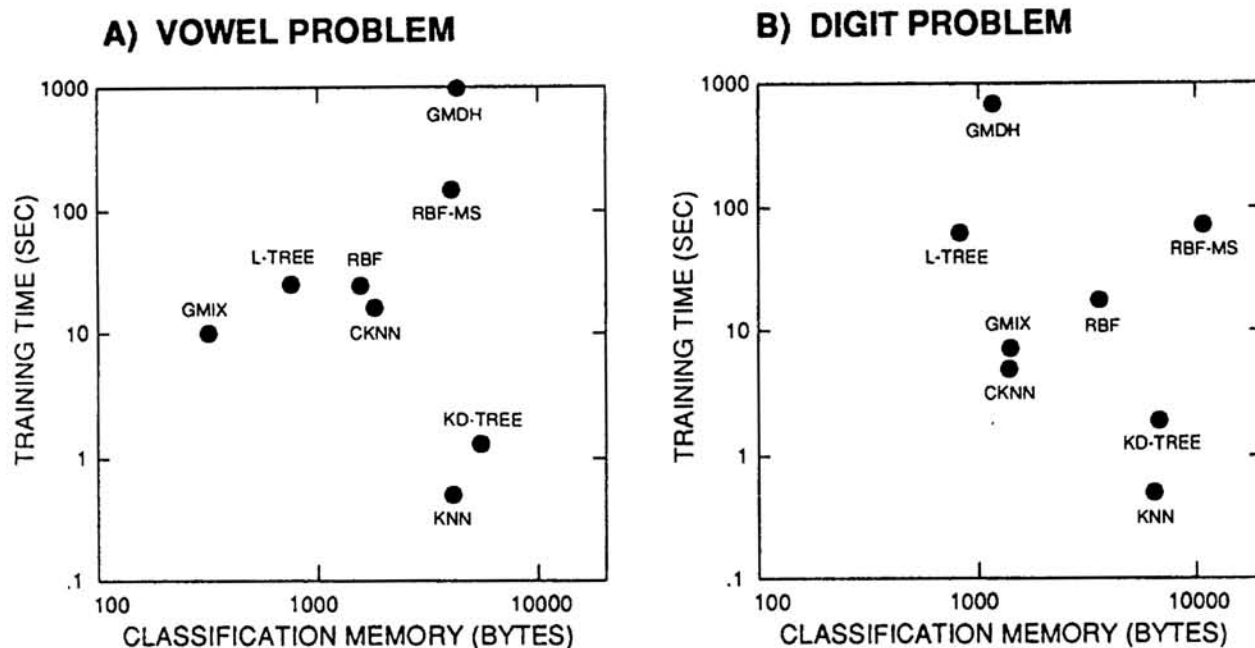

Figure 3: Training Time Versus Classification Memory Usage For All Classifiers On The (A) Vowel And (B) Digit Problems.

Figure 3 shows the relationship between training time (in CPU seconds measured on a Sun 3/110 with FPA) and classification memory usage (in bytes) for the different classifiers on the vowel and digit problems. On these problems, the KNN and KD-tree classifiers train quickly, but require large amounts of memory. The Gaussian mixture (GMIX) and linear tree (L-TREE) classifiers use little memory, but require more training time. The RBF and CKNN classifiers have intermediate memory and training time requirements. Due to the extra basis functions, the multiscale RBF (RBF-MS) classifier requires more training time and memory than the conventional RBF classifier. The GMDH classifier has intermediate memory requirements, but takes the longest to train. On average, the GMDH classifier takes 10 times longer to train than the RBF classifier, and 100 times longer than the KD tree classifier. In general, classifiers that use little memory require long training times, while those that train rapidly are not memory efficient.

Figure 4 shows the relationship between classification time (in CPU milliseconds

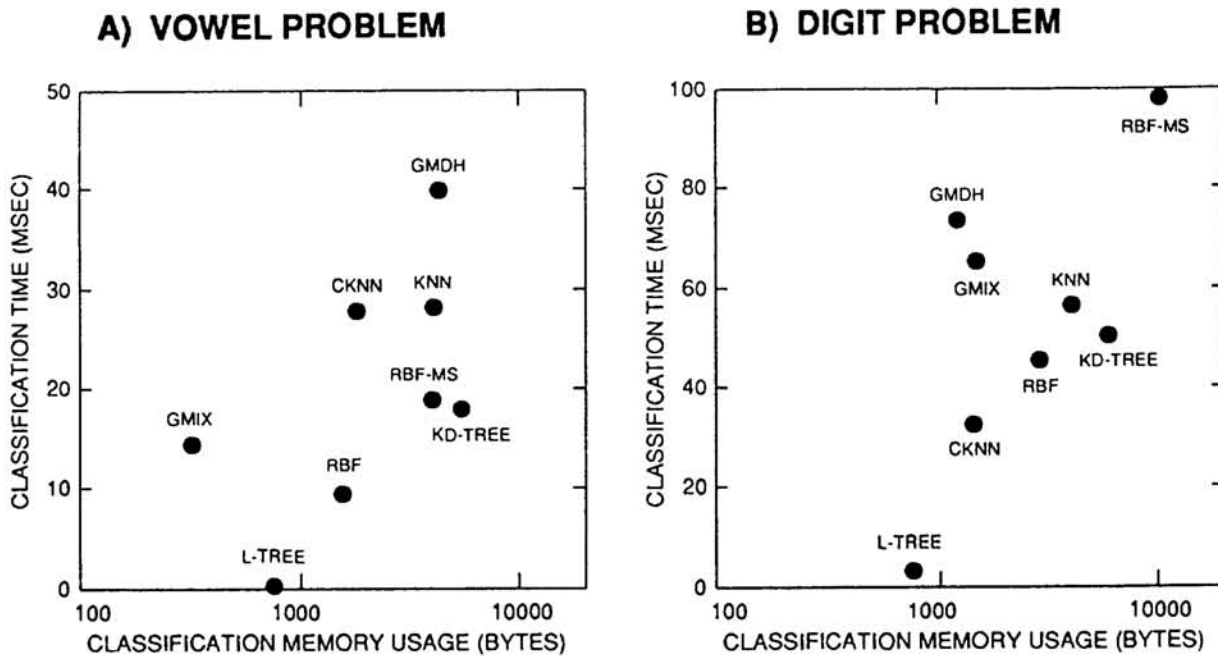

Figure 4: Classification Time Versus Classification Memory Usage For All Classifiers On The (A) Vowel And (B) Digit Problems.

for one pattern) and classification memory usage (in bytes) for the different classifiers on the vowel and digit problems. At one extreme, the linear tree classifier requires very little memory and classifies almost instantaneously. At the other, the GMDH classifier takes the longest to classify and requires a large amount of memory. Gaussian mixture and RBF classifiers are intermediate. On the vowel problem, the CKNN and the KD tree classifiers are faster than the conventional KNN classifier. On the digit problem, the CKNN classifier is faster than both the KD tree and KNN classifiers because of the greatly reduced number of stored patterns (15 out of 70). The speed up in search provided by the KD tree is greatly reduced for the digit problem due to the increase in input dimensionality. In general, the trend is for classification time to be proportional to the amount of classification memory. It is important to note, however, that trade-offs in performance characteristics depend on the particular problem and can vary for different implementations of the classifiers.

## 6   SUMMARY

Seven different neural network and conventional pattern classifiers were compared using artificial and speech recognition tasks. High order polynomial GMDH classifiers typically provided intermediate error rates and often required long training times and large amounts of memory. In addition, the decision regions formed did not generalize well to regions of the input space with little training data. Radial basis function classifiers generalized well in high dimensional spaces, and provided low error rates with training times that were much less than those of back-propagation classifiers (Lee and Lippmann, 1989). Gaussian mixture classifiers provided good performance when the numbers and types of mixtures were selected carefully to model class densities well. Linear tree classifiers were the most computationally ef-

ficient but performed poorly with high dimensionality inputs and when the number of training patterns was small. KD-tree classifiers reduced classification time by a factor of four over conventional KNN classifiers for low 2-input dimension problems. They provided little or no reduction in classification time for high 22-input dimension problems. Improved condensed KNN classifiers reduced memory requirements over conventional KNN classifiers by a factor of two to fifteen for all problems, without increasing the error rate significantly.

## 7  CONCLUSION

This and a previous study (Lee and Lippmann, 1989) explored the performance of 18 neural network, AI, and statistical pattern classifiers. Both studies demonstrated the need to carefully select and tune global parameters and the need to match classifier complexity to that of the training data using cross-validation and/or information theoretic approaches. Two new variants of existing classifiers (multi-scale RBF and improved versions of the CKNN classifier) were developed as part of this study. Classification error rates on speech problems in both studies were equivalent with most classifiers when classifiers were powerful enough to form minimum error decision regions, when sufficient training data was available, and when classifiers were carefully tuned. Practical classifier characteristics including training time, classification time, and memory usage, however, differed by orders of magnitude. These results suggest that the selection of a classifier for a particular task should be guided not so much by small differences in error rate, but by practical considerations concerning memory usage, ease of implementation, computational resources, and restrictions on training and classification times. Researchers should take time to understand the wide range of classifiers that are available and the practical tradeoffs that these classifiers provide.

### Acknowledgements

This work was sponsored by the Air Force Office of Scientific Research and the Department of the Air Force.

### References

Eric I. Chang and Richard P. Lippmann. Using Genetic Algorithms to Improve Pattern Classification Performance. In Lippmann, R. Moody, J., Touretzky, D., (Eds.) *Advances in Neural Information Processing Systems 3*, 1990.

Yuchun Lee. Classifiers: Adaptive modules in pattern recognition systems. Master's Thesis, Massachusetts Institute of Technology, Department of Electrical Engineering and Computer Science, Cambridge, MA, May 1989.

Yuchun Lee and R. P. Lippmann. Practical Characteristics of Neural Network and Conventional Pattern Classifiers on Artificial and Speech Problems. In D. Touretzky (Ed.) *Advances in Neural Information Processing Systems 2*, 168-177, 1989.

R. P. Lippmann. Pattern Classification Using Neural Networks. *IEEE Communications Magazine*, 27(27):47-54, 1989.

Kenney Ng. A Comparative Study of the Practical Characteristics of Neural Network and Conventional Pattern Classifiers. Master's Thesis, Massachusetts Institute of Technology, Department of Electrical Engineering and Computer Science, Cambridge, MA, May 1990.